# SELF-ORGANIZATION OF ASSOCIATIVE DATABASE AND ITS APPLICATIONS

**Hisashi Suzuki** and **Suguru Arimoto**
Osaka University, Toyonaka, Osaka 560, Japan

## ABSTRACT

An efficient method of self-organizing associative databases is proposed together with applications to robot eyesight systems. The proposed databases can associate any input with some output. In the first half part of discussion, an algorithm of self-organization is proposed. From an aspect of hardware, it produces a new style of neural network. In the latter half part, an applicability to handwritten letter recognition and that to an autonomous mobile robot system are demonstrated.

## INTRODUCTION

Let a mapping $f : X \to Y$ be given. Here, $X$ is a finite or infinite set, and $Y$ is another finite or infinite set. A learning machine observes any set of pairs $(x, y)$ sampled randomly from $X \times Y$. ($X \times Y$ means the Cartesian product of $X$ and $Y$.) And, it computes some estimate $\hat{f} : X \to Y$ of $f$ to make small, the estimation error in some measure.

Usually we say that: the faster the decrease of estimation error with increase of the number of samples, the better the learning machine. However, such expression on performance is incomplete. Since, it lacks consideration on the candidates of $f$ of $\hat{f}$ assumed preliminarily. Then, how should we find out good learning machines? To clarify this conception, let us discuss for a while on some types of learning machines. And, let us advance the understanding of the self-organization of associative database.

· Parameter Type
An ordinary type of learning machine assumes an equation relating $x$'s and $y$'s with parameters being indefinite, namely, a structure of $f$. It is equivalent to define implicitly a set $\hat{F}$ of candidates of $\hat{f}$. ($\hat{F}$ is some subset of mappings from $X$ to $Y$.) And, it computes values of the parameters based on the observed samples. We call such type a parameter type.

For a learning machine defined well, if $\hat{F} \ni f$, $\hat{f}$ approaches $f$ as the number of samples increases. In the alternative case, however, some estimation error remains eternally. Thus, a problem of designing a learning machine returns to find out a proper structure of $f$ in this sense.

On the other hand, the assumed structure of $f$ is demanded to be as compact as possible to achieve a fast learning. In other words, the number of parameters should be small. Since, if the parameters are few, some $\hat{f}$ can be uniquely determined even though the observed samples are few. However, this demand of being proper contradicts to that of being compact. Consequently, in the parameter type, the better the compactness of the assumed structure that is proper, the better the learning machine. This is the most elementary conception when we design learning machines.

· Universality and Ordinary Neural Networks
Now suppose that a sufficient knowledge on $f$ is given though $f$ itself is unknown. In this case, it is comparatively easy to find out proper and compact structures of $f$. In the alternative case, however, it is sometimes difficult. A possible solution is to give up the compactness and assume an almighty structure that can cover various $f$'s. A combination of some orthogonal bases of the infinite dimension is such a structure. Neural networks[1,2] are its approximations obtained by truncating finitely the dimension for implementation.

A main topic in designing neural networks is to establish such desirable structures of $f$. This work includes developing practical procedures that compute values of coefficients from the observed samples. Such discussions are flourishing since 1980 while many efficient methods have been proposed. Recently, even hardware units computing coefficients in parallel for speed-up are sold, e.g., ANZA, Mark III, Odyssey and $\Sigma$-1.

Nevertheless, in neural networks, there always exists a danger of some error remaining eternally in estimating $f$. Precisely speaking, suppose that a combination of the bases of a finite number can define a structure of $f$ essentially. In other words, suppose that $\hat{F} \ni f$, or $f$ is located near $\hat{F}$. In such case, the estimation error is none or negligible. However, if $f$ is distant from $\hat{F}$, the estimation error never becomes negligible. Indeed, many researches report that the following situation appears when $f$ is too complex. Once the estimation error converges to some value ($> 0$) as the number of samples increases, it decreases hardly even though the dimension is heighten. This property sometimes is a considerable defect of neural networks.

· Recursive Type

The recursive type is founded on another methodology of learning that should be as follows. At the initial stage of no sample, the set $\hat{F}_0$ (instead of notation $\hat{F}$) of candidates of $\hat{f}$ equals to the set of all mappings from $X$ to $Y$. After observing the first sample $(x_1, y_1) \in X \times Y$, $\hat{F}_0$ is reduced to $\hat{F}_1$ so that $\hat{f}(x_1) = y_1$ for any $\hat{f} \in \hat{F}$. After observing the second sample $(x_2, y_2) \in X \times Y$, $\hat{F}_1$ is further reduced to $\hat{F}_2$ so that $\hat{f}(x_1) = y_1$ and $\hat{f}(x_2) = y_2$ for any $\hat{f} \in \hat{F}$. Thus, the candidate set $\hat{F}$ becomes gradually small as observation of samples proceeds. The $\hat{f}$ after observing $i$-samples, which we write $\hat{f}_i$, is one of the most likelihood estimation of $f$ selected in $\hat{F}_i$. Hence, contrarily to the parameter type, the recursive type guarantees surely that $\hat{f}$ approaches to $f$ as the number of samples increases.

The recursive type, if observes a sample $(x_i, y_i)$, rewrites values $\hat{f}_{i-1}(\tilde{x})$'s to $\hat{f}_i(\tilde{x})$'s for some $\tilde{x}$'s correlated to the sample. Hence, this type has an architecture composed of a rule for rewriting and a free memory space. Such architecture forms naturally a kind of database that builds up management systems of data in a self-organizing way. However, this database differs from ordinary ones in the following sense. It does not only record the samples already observed, but computes some estimation of $f(x)$ for any $x \in X$. We call such database an associative database.

The first subject in constructing associative databases is how we establish the rule for rewriting. For this purpose, we adapt a measure called the dissimilarity. Here, a dissimilarity means a mapping $d : X \times X \to \{\text{reals} > 0\}$ such that for any $(x, \tilde{x}) \in X \times X$, $d(x, \tilde{x}) > 0$ whenever $f(x) \neq f(\tilde{x})$. However, it is not necessarily defined with a single formula. It is definable with, for example, a collection of rules written in forms of "if $\cdots$ then $\cdots$."

The dissimilarity $d$ defines a structure of $f$ locally in $X \times Y$. Hence, even though the knowledge on $f$ is imperfect, we can reflect it on $d$ in some heuristic way. Hence, contrarily to neural networks, it is possible to accelerate the speed of learning by establishing $d$ well. Especially, we can easily find out simple $d$'s for those $f$'s which process analogically information like a human. (See the applications in this paper.) And, for such $f$'s, the recursive type shows strongly its effectiveness.

We denote a sequence of observed samples by $(x_1, y_1), (x_2, y_2), \cdots$. One of the simplest constructions of associative databases $\hat{f}_i$ after observing $i$-samples ($i = 1, 2, \cdots$) is as follows.

**Algorithm 1.** At the initial stage, let $S_0$ be the empty set. For every $i = 1, 2, \cdots$, let $\hat{f}_{i-1}(x)$ for any $x \in X$ equal some $y^*$ such that $(x^*, y^*) \in S_{i-1}$ and

$$d(x, x^*) = \min_{(\tilde{x}, \tilde{y}) \in S_{i-1}} d(x, \tilde{x}) . \tag{1}$$

Furthermore, add $(x_i, y_i)$ to $S_{i-1}$ to produce $S_i$, i.e., $S_i = S_{i-1} \cup \{(x_i, y_i)\}$.

Another version improved to economize the memory is as follows.

**Algorithm 2.** At the initial stage, let $S_0$ be composed of an arbitrary element in $X \times Y$. For every $i = 1, 2, \cdots$, let $\hat{f}_{i-1}(x)$ for any $x \in X$ equal some $y^*$ such that $(x^*, y^*) \in S_{i-1}$ and

$$d(x, x^*) = \min_{(\tilde{x}, \tilde{y}) \in S_{i-1}} d(x, \tilde{x}) .$$

Furthermore, if $\hat{f}_{i-1}(x_i) \neq y_i$ then let $S_i = S_{i-1}$, or add $(x_i, y_i)$ to $S_{i-1}$ to produce $S_i$, i.e., $S_i = S_{i-1} \cup \{(x_i, y_i)\}$.

In either construction, $\hat{f}_i$ approaches to $f$ as $i$ increases. However, the computation time grows proportionally to the size of $S_i$. The second subject in constructing associative databases is what addressing rule we should employ to economize the computation time. In the subsequent chapters, a construction of associative database for this purpose is proposed. It manages data in a form of binary tree.

## SELF-ORGANIZATION OF ASSOCIATIVE DATABASE

Given a sample sequence $(x_1, y_1), (x_2, y_2), \cdots$, the algorithm for constructing associative database is as follows.

**Algorithm 3.**
Step 1(Initialization): Let $(x[\text{root}], y[\text{root}]) = (x_1, y_1)$. Here, $x[\cdot]$ and $y[\cdot]$ are variables assigned for respective nodes to memorize data. Furthermore, let $t = 1$.

Step 2: Increase $t$ by 1, and put $x_t$ in. After reset a pointer $n$ to the root, repeat the following until $n$ arrives at some terminal node, i.e., leaf.

Notations $\acute{n}$ and $\grave{n}$ mean the descendant nodes of $n$. If $d(x_t, x[\acute{n}]) \leq d(x_t, x[\grave{n}])$, let $n = \acute{n}$. Otherwise, let $n = \grave{n}$.

Step 3: Display $y[n]$ as the related information. Next, put $y_t$ in. If $y[n] = y_t$, back to step 2. Otherwise, first establish new descendant nodes $\acute{n}$ and $\grave{n}$. Secondly, let

$$(x[\acute{n}], y[\acute{n}]) = (x[n], y[n]), \tag{2}$$
$$(x[\grave{n}], y[\grave{n}]) = (x_t, y_t). \tag{3}$$

Finally, back to step 2. Here, the loop of step 2–3 can be stopped at any time and also can be continued.

Now, suppose that gate elements, namely, artificial "synapses" that play the role of branching by $d$ are prepared. Then, we obtain a new style of neural network with gate elements being randomly connected by this algorithm.

## LETTER RECOGNITION

Recently, the vertical slitting method for recognizing typographic English letters[3], the elastic matching method for recognizing handwritten discrete English letters[4], the global training and fuzzy logic search method for recognizing Chinese characters written in square style[5], etc. are published. The self-organization of associative database realizes the recognition of handwritten continuous English letters.

*I been in some meetings where the tab
contorted and the chairs knotted and the w
one another till you could of wrung swea
in meetings where they kept talking about a*

**Fig. 1. Source document.**

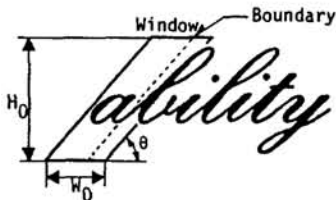

**Fig. 2. Windowing.**

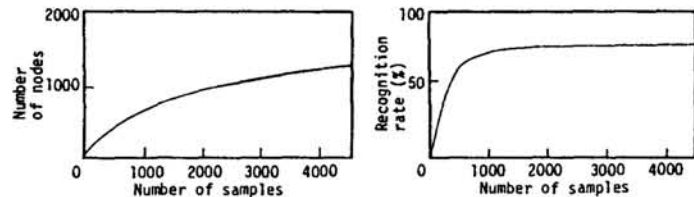

**Fig. 3. An experiment result.**

An image scanner takes a document image (**Fig. 1**). The letter recognizer uses a parallelogram window that at least can cover the maximal letter (**Fig. 2**), and processes the sequence of letters while shifting the window. That is, the recognizer scans a word in a slant direction. And, it places the window so that its left vicinity may be on the first black point detected. Then, the window catches a letter and some part of the succeeding letter. If recognition of the head letter is performed, its end position, namely, the boundary line between two letters becomes known. Hence, by starting the scanning from this boundary and repeating the above operations, the recognizer accomplishes recursively the task. Thus the major problem comes to identifying the head letter in the window.

Considering it, we define the following.

- Regard window images as $x$'s, and define $X$ accordingly.

- For a $(x, \tilde{x}) \in X \times X$, denote by $\tilde{B}$ a black point in the left area from the boundary on window image $\tilde{x}$. Project each $\tilde{B}$ onto window image $x$. Then, measure the Euclidean distance $\delta$ between $\tilde{B}$ and a black point $B$ on $x$ being the closest to $\tilde{B}$. Let $d(x, \tilde{x})$ be the summation of $\delta$'s for all black points $\tilde{B}$'s on $\tilde{x}$ divided by the number of $\tilde{B}$'s.

- Regard couples of the "reading" and the position of boundary as $y$'s, and define $Y$ accordingly.

An operator teaches the recognizer in interaction the relation between window image and reading&boundary with algorithm 3. Precisely, if the recalled reading is incorrect, the operator teaches a correct reading via the console. Moreover, if the boundary position is incorrect, he teaches a correct position via the mouse.

**Fig. 1** shows partially a document image used in this experiment. **Fig. 3** shows the change of the number of nodes and that of the recognition rate defined as the relative frequency of correct answers in the past 1000 trials. Specifications of the window are height = 20dot, width = 10dot, and slant angular = 68deg. In this example, the levels of tree were distributed in 6–19 at time 4000 and the recognition rate converged to about 74%. Experimentally, the recognition rate converges to about 60–85% in most cases, and to 95% at a rare case. However, it does not attain 100% since, e.g., "c" and "e" are not distinguishable because of excessive fluctuation in writing. If the consistency of the $x, y$-relation is not assured like this, the number of nodes increases endlessly (cf. **Fig. 3**). Hence, it is clever to stop the learning when the recognition rate attains some upper limit. To improve further the recognition rate, we must consider the spelling of words. It is one of future subjects.

## OBSTACLE AVOIDING MOVEMENT

Various systems of camera type autonomous mobile robot are reported flourishingly[6-10]. The system made up by the authors (**Fig. 4**) also belongs to this category. Now, in mathematical methodologies, we solve usually the problem of obstacle avoiding movement as a cost minimization problem under some cost criterion established artificially. Contrarily, the self-organization of associative database reproduces faithfully the cost criterion of an operator. Therefore, motion of the robot after learning becomes very natural.

Now, the length, width and height of the robot are all about 0.7m, and the weight is about 30kg. The visual angle of camera is about 55deg. The robot has the following three factors of motion. It turns less than ±30deg, advances less than 1m, and controls speed less than 3km/h. The experiment was done on the passageway of width 2.5m inside a building which the authors' laboratories exist in (**Fig. 5**). Because of an experimental intention, we arrange boxes, smoking stands, gas cylinders, stools, handcarts, etc. on the passage way at random. We let the robot take an image through the camera, recall a similar image, and trace the route preliminarily recorded on it. For this purpose, we define the following.

- Let the camera face 28deg downward to take an image, and process it through a low pass filter. Scanning vertically the filtered image from the bottom to the top, search the first point $C$ where the luminance changes excessively. Then, substitute all points from the bottom to $C$ for white, and all points from $C$ to the top for black (**Fig. 6**). (If no obstacle exists just in front of the robot, the white area shows the "free" area where the robot can move around.) Regard binary 32 × 32dot images processed thus as $x$'s, and define $X$ accordingly.

- For every $(x,\tilde{x}) \in X \times X$, let $d(x,\tilde{x})$ be the number of black points on the exclusive-or image between $x$ and $\tilde{x}$.

- Regard as $y$'s the images obtained by drawing routes on images $x$'s, and define $Y$ accordingly.

The robot superimposes, on the current camera image $x$, the route recalled for $x$, and inquires the operator instructions. The operator judges subjectively whether the suggested route is appropriate or not. In the negative answer, he draws a desirable route on $x$ with the mouse to teach a new $y$ to the robot. This operation defines implicitly a sample sequence of $(x,y)$ reflecting the cost criterion of the operator.

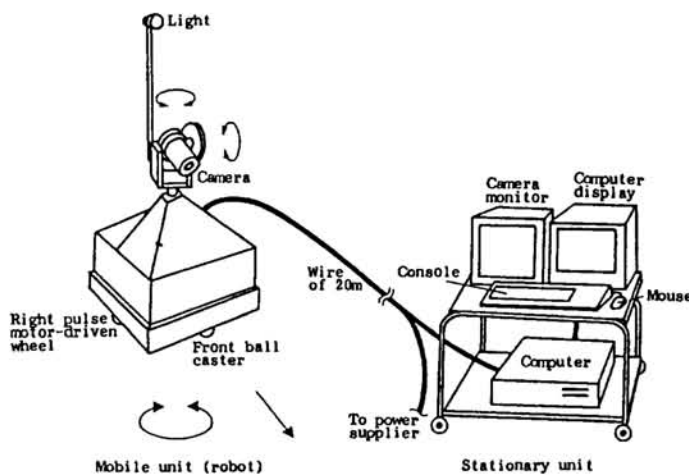

**Fig. 4. Configuration of autonomous mobile robot system.**

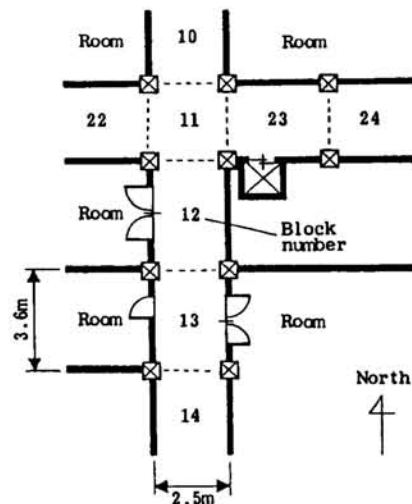

**Fig. 5. Experimental environment.**

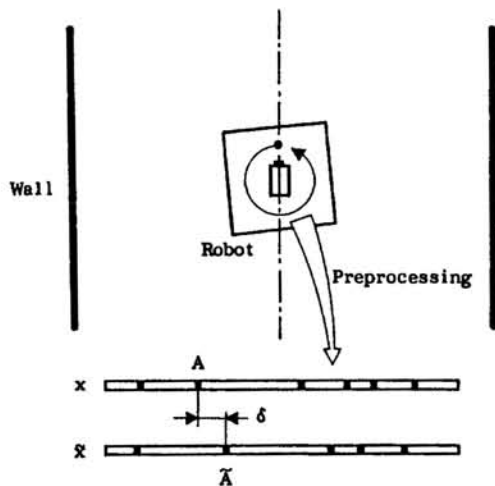

Fig. 6. Processing for
obstacle avoiding movement.

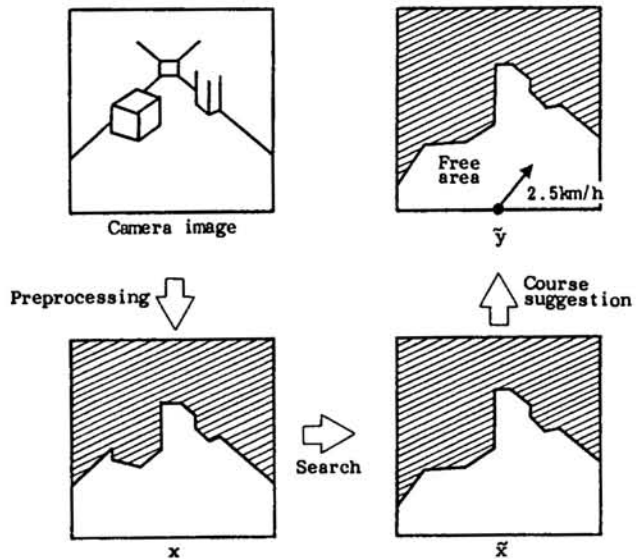

Fig. 7. Processing for
position identification.

We define the satisfaction rate by the relative frequency of acceptable suggestions of route in the past 100 trials. In a typical experiment, the change of satisfaction rate showed a similar tendency to **Fig. 3**, and it attains about 95% around time 800. Here, notice that the rest 5% does not mean directly the percentage of collision. (In practice, we prevent the collision by adopting some supplementary measure.) At time 800, the number of nodes was 145, and the levels of tree were distributed in 6–17.

The proposed method reflects delicately various characters of operator. For example, a robot trained by an operator O moves slowly with enough space against obstacles while one trained by another operator O' brushes quickly against obstacles. This fact gives us a hint on a method of printing "characters" into machines.

## POSITION IDENTIFICATION

The robot can identify its position by recalling a similar landscape with the position data to a camera image. For this purpose, in principle, it suffices to regard camera images and position data as $x$'s and $y$'s, respectively. However, the memory capacity is finite in actual computers. Hence, we cannot but compress the camera images at a slight loss of information. Such compression is admittable as long as the precision of position identification is in an acceptable area. Thus, the major problem comes to find out some suitable compression method.

In the experimental environment (**Fig. 5**), juts are on the passageway at intervals of $3.6m$, and each section between adjacent juts has at most one door. The robot identifies roughly from a surrounding landscape which section itself places in. And, it uses temporarily a triangular surveying technique if an exact measure is necessary. To realize the former task, we define the following.

- Turn the camera to take a panorama image of 360deg. Scanning horizontally the center line, substitute the points where the luminance excessively changes for black and the other points for white (**Fig. 7**). Regard binary 360dot line images processed thus as $x$'s, and define $X$ accordingly.

- For every $(x, \tilde{x}) \in X \times X$, project each black point $\tilde{A}$ on $\tilde{x}$ onto $x$. And, measure the Euclidean distance $\delta$ between $\tilde{A}$ and a black point $A$ on $x$ being the closest to $\tilde{A}$. Let the summation of $\delta$ be $S$. Similarly, calculate $\tilde{S}$ by exchanging the roles of $x$ and $\tilde{x}$. Denoting the numbers of $A$'s and $\tilde{A}$'s respectively by $n$ and $\tilde{n}$, define

$$d(x, \tilde{x}) = \frac{1}{2}\left(\frac{S}{n} + \frac{\tilde{S}}{\tilde{n}}\right). \tag{4}$$

- Regard positive integers labeled on sections as $y$'s (cf. **Fig. 5**), and define $Y$ accordingly.

In the learning mode, the robot checks exactly its position with a counter that is reset periodically by the operator. The robot runs arbitrarily on the passageways within 18m area and learns the relation between landscapes and position data. (Position identification beyond 18m area is achieved by crossing plural databases one another.) This task is automatic excepting the periodic reset of counter, namely, it is a kind of learning without teacher.

We define the identification rate by the relative frequency of correct recalls of position data in the past 100 trials. In a typical example, it converged to about 83% around time 400. At time 400, the number of levels was 202, and the levels of tree were distributed in 5–22. Since the identification failures of 17% can be rejected by considering the trajectory, no problem arises in practical use. In order to improve the identification rate, the compression ratio of camera images must be loosened. Such possibility depends on improvement of the hardware in the future.

**Fig. 8** shows an example of actual motion of the robot based on the database for obstacle avoiding movement and that for position identification. This example corresponds to a case of moving from 14 to 23 in **Fig. 5**. Here, the time interval per frame is about 40sec.

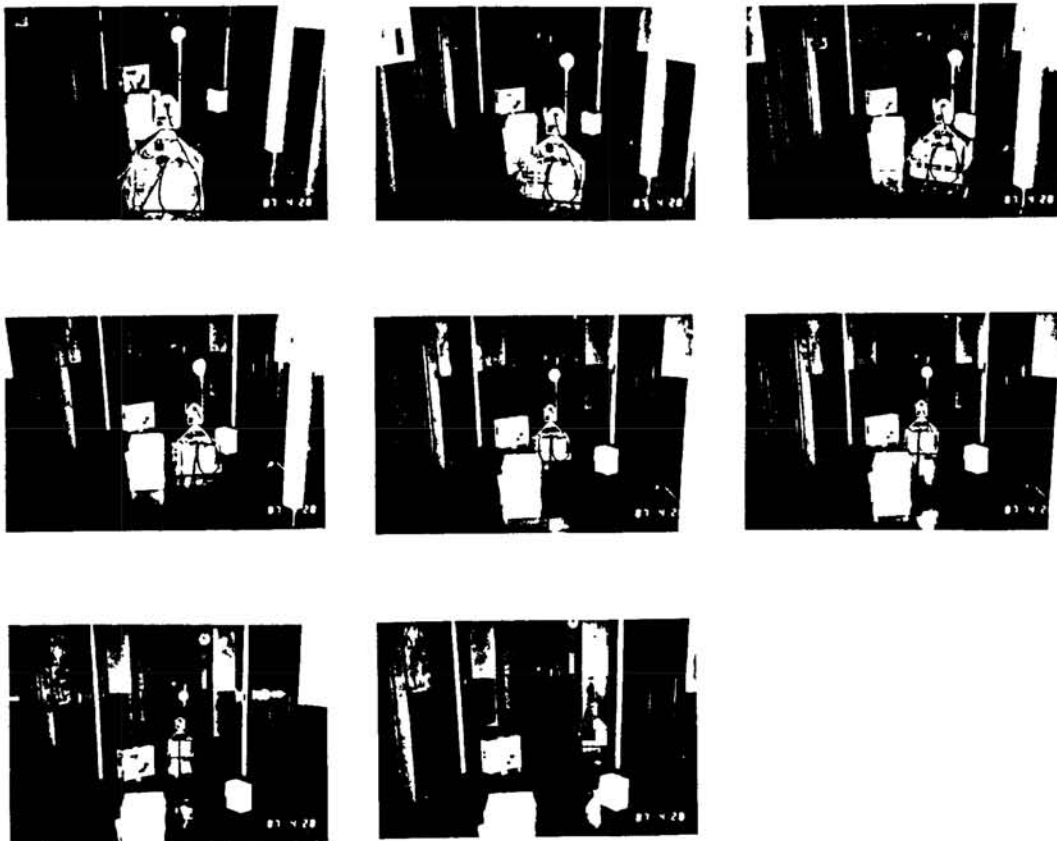

**Fig. 8. Actual motion of the robot.**

## CONCLUSION

A method of self-organizing associative databases was proposed with the application to robot eyesight systems. The machine decomposes a global structure unknown into a set of local structures known and learns universally any input-output response. This framework of problem implies a wide application area other than the examples shown in this paper.

A defect of the algorithm 3 of self-organization is that the tree is balanced well only for a subclass of structures of $f$. A subject imposed us is to widen the class. A probable solution is to abolish the addressing rule depending directly on values of $d$ and, instead, to establish another rule depending on the distribution function of values of $d$. It is now under investigation.

## REFERENCES

1. Hopfield, J. J. and D. W. Tank, "Computing with Neural Circuit: A Model," Science **233** (1986), pp. 625–633.

2. Rumelhart, D. E. et al., "Learning Representations by Back-Propagating Errors," Nature **323** (1986), pp. 533–536.

3. Hull, J. J., "Hypothesis Generation in a Computational Model for Visual Word Recognition," IEEE Expert, **Fall** (1986), pp. 63–70.

4. Kurtzberg, J. M., "Feature Analysis for Symbol Recognition by Elastic Matching," IBM J. Res. Develop. **31-1** (1987), pp. 91–95.

5. Wang, Q. R. and C. Y. Suen, "Large Tree Classifier with Heuristic Search and Global Training," IEEE Trans. Pattern. Anal. & Mach. Intell. **PAMI 9-1** (1987) pp. 91–102.

6. Brooks, R. A. et al, "Self Calibration of Motion and Stereo Vision for Mobile Robots," 4th Int. Symp. of Robotics Research (1987), pp. 267–276.

7. Goto, Y. and A. Stentz, "The CMU System for Mobile Robot Navigation," 1987 IEEE Int. Conf. on Robotics & Automation (1987), pp. 99–105.

8. Madarasz, R. et al., "The Design of an Autonomous Vehicle for the Disabled," IEEE Jour. of Robotics & Automation **RA 2-3** (1986), pp. 117–125.

9. Triendl, E. and D. J. Kriegman, "Stereo Vision and Navigation within Buildings," 1987 IEEE Int. Conf. on Robotics & Automation (1987), pp. 1725–1730.

10. Turk, M. A. et al., "Video Road-Following for the Autonomous Land Vehicle," 1987 IEEE Int. Conf. on Robotics & Automation (1987), pp. 273–279.
